# Algorithms for Non-negative Matrix Factorization

**Daniel D. Lee***
*Bell Laboratories
Lucent Technologies
Murray Hill, NJ 07974

**H. Sebastian Seung***[†]
†Dept. of Brain and Cog. Sci.
Massachusetts Institute of Technology
Cambridge, MA 02138

## Abstract

Non-negative matrix factorization (NMF) has previously been shown to be a useful decomposition for multivariate data. Two different multiplicative algorithms for NMF are analyzed. They differ only slightly in the multiplicative factor used in the update rules. One algorithm can be shown to minimize the conventional least squares error while the other minimizes the generalized Kullback-Leibler divergence. The monotonic convergence of both algorithms can be proven using an auxiliary function analogous to that used for proving convergence of the Expectation-Maximization algorithm. The algorithms can also be interpreted as diagonally rescaled gradient descent, where the rescaling factor is optimally chosen to ensure convergence.

## 1  Introduction

Unsupervised learning algorithms such as principal components analysis and vector quantization can be understood as factorizing a data matrix subject to different constraints. Depending upon the constraints utilized, the resulting factors can be shown to have very different representational properties. Principal components analysis enforces only a weak orthogonality constraint, resulting in a very distributed representation that uses cancellations to generate variability [1, 2]. On the other hand, vector quantization uses a hard winner-take-all constraint that results in clustering the data into mutually exclusive prototypes [3].

We have previously shown that nonnegativity is a useful constraint for matrix factorization that can learn a parts representation of the data [4, 5]. The nonnegative basis vectors that are learned are used in distributed, yet still sparse combinations to generate expressiveness in the reconstructions [6, 7]. In this submission, we analyze in detail two numerical algorithms for learning the optimal nonnegative factors from data.

## 2  Non-negative matrix factorization

We formally consider algorithms for solving the following problem:

> **Non-negative matrix factorization (NMF)** Given a non-negative matrix $V$, find non-negative matrix factors $W$ and $H$ such that:

$$V \approx WH \tag{1}$$

NMF can be applied to the statistical analysis of multivariate data in the following manner. Given a set of of multivariate $n$-dimensional data vectors, the vectors are placed in the columns of an $n \times m$ matrix $V$ where $m$ is the number of examples in the data set. This matrix is then approximately factorized into an $n \times r$ matrix $W$ and an $r \times m$ matrix $H$. Usually $r$ is chosen to be smaller than $n$ or $m$, so that $W$ and $H$ are smaller than the original matrix $V$. This results in a compressed version of the original data matrix.

What is the significance of the approximation in Eq. (1)? It can be rewritten column by column as $v \approx Wh$, where $v$ and $h$ are the corresponding columns of $V$ and $H$. In other words, each data vector $v$ is approximated by a linear combination of the columns of $W$, weighted by the components of $h$. Therefore $W$ can be regarded as containing a basis that is optimized for the linear approximation of the data in $V$. Since relatively few basis vectors are used to represent many data vectors, good approximation can only be achieved if the basis vectors discover structure that is latent in the data.

The present submission is not about applications of NMF, but focuses instead on the technical aspects of finding non-negative matrix factorizations. Of course, other types of matrix factorizations have been extensively studied in numerical linear algebra, but the non-negativity constraint makes much of this previous work inapplicable to the present case [8].

Here we discuss two algorithms for NMF based on iterative updates of $W$ and $H$. Because these algorithms are easy to implement and their convergence properties are guaranteed, we have found them very useful in practical applications. Other algorithms may possibly be more efficient in overall computation time, but are more difficult to implement and may not generalize to different cost functions. Algorithms similar to ours where only one of the factors is adapted have previously been used for the deconvolution of emission tomography and astronomical images [9, 10, 11, 12].

At each iteration of our algorithms, the new value of $W$ or $H$ is found by multiplying the current value by some factor that depends on the quality of the approximation in Eq. (1). We prove that the quality of the approximation improves monotonically with the application of these multiplicative update rules. In practice, this means that repeated iteration of the update rules is guaranteed to converge to a locally optimal matrix factorization.

## 3 Cost functions

To find an approximate factorization $V \approx WH$, we first need to define cost functions that quantify the quality of the approximation. Such a cost function can be constructed using some measure of distance between two non-negative matrices $A$ and $B$. One useful measure is simply the square of the Euclidean distance between $A$ and $B$ [13],

$$||A - B||^2 = \sum_{ij}(A_{ij} - B_{ij})^2 \tag{2}$$

This is lower bounded by zero, and clearly vanishes if and only if $A = B$.

Another useful measure is

$$D(A||B) = \sum_{ij} \left( A_{ij} \log \frac{A_{ij}}{B_{ij}} - A_{ij} + B_{ij} \right) \tag{3}$$

Like the Euclidean distance this is also lower bounded by zero, and vanishes if and only if $A = B$. But it cannot be called a "distance", because it is not symmetric in $A$ and $B$, so we will refer to it as the "divergence" of $A$ from $B$. It reduces to the Kullback-Leibler divergence, or relative entropy, when $\sum_{ij} A_{ij} = \sum_{ij} B_{ij} = 1$, so that $A$ and $B$ can be regarded as normalized probability distributions.

We now consider two alternative formulations of NMF as optimization problems:

**Problem 1** *Minimize $||V - WH||^2$ with respect to $W$ and $H$, subject to the constraints $W, H \geq 0$.*

**Problem 2** *Minimize $D(V||WH)$ with respect to $W$ and $H$, subject to the constraints $W, H \geq 0$.*

Although the functions $||V - WH||^2$ and $D(V||WH)$ are convex in $W$ only or $H$ only, they are not convex in both variables together. Therefore it is unrealistic to expect an algorithm to solve Problems 1 and 2 in the sense of finding global minima. However, there are many techniques from numerical optimization that can be applied to find local minima.

Gradient descent is perhaps the simplest technique to implement, but convergence can be slow. Other methods such as conjugate gradient have faster convergence, at least in the vicinity of local minima, but are more complicated to implement than gradient descent [8]. The convergence of gradient based methods also have the disadvantage of being very sensitive to the choice of step size, which can be very inconvenient for large applications.

## 4 Multiplicative update rules

We have found that the following "multiplicative update rules" are a good compromise between speed and ease of implementation for solving Problems 1 and 2.

**Theorem 1** *The Euclidean distance $||V - WH||$ is nonincreasing under the update rules*

$$H_{a\mu} \leftarrow H_{a\mu} \frac{(W^T V)_{a\mu}}{(W^T W H)_{a\mu}} \qquad W_{ia} \leftarrow W_{ia} \frac{(V H^T)_{ia}}{(W H H^T)_{ia}} \qquad (4)$$

*The Euclidean distance is invariant under these updates if and only if $W$ and $H$ are at a stationary point of the distance.*

**Theorem 2** *The divergence $D(V||WH)$ is nonincreasing under the update rules*

$$H_{a\mu} \leftarrow H_{a\mu} \frac{\sum_i W_{ia} V_{i\mu}/(WH)_{i\mu}}{\sum_k W_{ka}} \qquad W_{ia} \leftarrow W_{ia} \frac{\sum_\mu H_{a\mu} V_{i\mu}/(WH)_{i\mu}}{\sum_\nu H_{a\nu}} \qquad (5)$$

*The divergence is invariant under these updates if and only if $W$ and $H$ are at a stationary point of the divergence.*

Proofs of these theorems are given in a later section. For now, we note that each update consists of multiplication by a factor. In particular, it is straightforward to see that this multiplicative factor is unity when $V = WH$, so that perfect reconstruction is necessarily a fixed point of the update rules.

## 5 Multiplicative versus additive update rules

It is useful to contrast these multiplicative updates with those arising from gradient descent [14]. In particular, a simple additive update for $H$ that reduces the squared distance can be written as

$$H_{a\mu} \leftarrow H_{a\mu} + \eta_{a\mu} \left[ (W^T V)_{a\mu} - (W^T W H)_{a\mu} \right]. \qquad (6)$$

If $\eta_{a\mu}$ are all set equal to some small positive number, this is equivalent to conventional gradient descent. As long as this number is sufficiently small, the update should reduce $||V - WH||$.

Now if we diagonally rescale the variables and set

$$\eta_{a\mu} = \frac{H_{a\mu}}{(W^T W H)_{a\mu}}, \tag{7}$$

then we obtain the update rule for $H$ that is given in Theorem 1. Note that this rescaling results in a multiplicative factor with the positive component of the gradient in the denominator and the absolute value of the negative component in the numerator of the factor.

For the divergence, diagonally rescaled gradient descent takes the form

$$H_{a\mu} \leftarrow H_{a\mu} + \eta_{a\mu} \left[ \sum_i W_{ia} \frac{V_{i\mu}}{(WH)_{i\mu}} - \sum_i W_{ia} \right]. \tag{8}$$

Again, if the $\eta_{a\mu}$ are small and positive, this update should reduce $D(V \| WH)$. If we now set

$$\eta_{a\mu} = \frac{H_{a\mu}}{\sum_i W_{ia}}, \tag{9}$$

then we obtain the update rule for $H$ that is given in Theorem 2. This rescaling can also be interpreted as a multiplicative rule with the positive component of the gradient in the denominator and negative component as the numerator of the multiplicative factor.

Since our choices for $\eta_{a\mu}$ are not small, it may seem that there is no guarantee that such a rescaled gradient descent should cause the cost function to decrease. Surprisingly, this is indeed the case as shown in the next section.

## 6   Proofs of convergence

To prove Theorems 1 and 2, we will make use of an auxiliary function similar to that used in the Expectation-Maximization algorithm [15, 16].

**Definition 1** $G(h, h')$ *is an* auxiliary function *for* $F(h)$ *if the conditions*

$$G(h, h') \geq F(h), \qquad G(h, h) = F(h) \tag{10}$$

*are satisfied.*

The auxiliary function is a useful concept because of the following lemma, which is also graphically illustrated in Fig. 1.

**Lemma 1** *If $G$ is an auxiliary function, then $F$ is nonincreasing under the update*

$$h^{t+1} = \arg \min_h G(h, h^t) \tag{11}$$

**Proof:** $F(h^{t+1}) \leq G(h^{t+1}, h^t) \leq G(h^t, h^t) = F(h^t)$ ∎

Note that $F(h^{t+1}) = F(h^t)$ only if $h^t$ is a local minimum of $G(h, h^t)$. If the derivatives of $F$ exist and are continuous in a small neighborhood of $h^t$, this also implies that the derivatives $\nabla F(h^t) = 0$. Thus, by iterating the update in Eq. (11) we obtain a sequence of estimates that converge to a local minimum $h_{\min} = \arg \min_h F(h)$ of the objective function:

$$F(h_{\min}) \leq ...F(h^{t+1}) \leq F(h^t)... \leq F(h_2) \leq F(h_1) \leq F(h_0). \tag{12}$$

We will show that by defining the appropriate auxiliary functions $G(h, h^t)$ for both $\| V - WH \|$ and $D(V, WH)$, the update rules in Theorems 1 and 2 easily follow from Eq. (11).

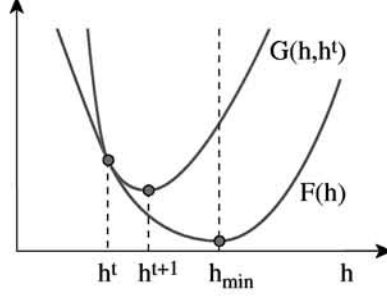

Figure 1: Minimizing the auxiliary function $G(h, h^t) \geq F(h)$ guarantees that $F(h^{t+1}) \leq F(h^t)$ for $h^{n+1} = \arg\min_h G(h, h^t)$.

**Lemma 2** *If $K(h^t)$ is the diagonal matrix*

$$K_{ab}(h^t) = \delta_{ab}(W^T W h^t)_a / h_a^t \tag{13}$$

*then*

$$G(h, h^t) = F(h^t) + (h - h^t)^T \nabla F(h^t) + \frac{1}{2}(h - h^t)^T K(h^t)(h - h^t) \tag{14}$$

*is an auxiliary function for*

$$F(h) = \frac{1}{2} \sum_i (v_i - \sum_a W_{ia} h_a)^2 \tag{15}$$

**Proof:** Since $G(h, h) = F(h)$ is obvious, we need only show that $G(h, h^t) \geq F(h)$. To do this, we compare

$$F(h) = F(h^t) + (h - h^t)^T \nabla F(h^t) + \frac{1}{2}(h - h^t)^T (W^T W)(h - h^t) \tag{16}$$

with Eq. (14) to find that $G(h, h^t) \geq F(h)$ is equivalent to

$$0 \leq (h - h^t)^T [K(h^t) - W^T W](h - h^t) \tag{17}$$

To prove positive semidefiniteness, consider the matrix[1]:

$$M_{ab}(h^t) = h_a^t (K(h^t) - W^T W)_{ab} h_b^t \tag{18}$$

which is just a rescaling of the components of $K - W^T W$. Then $K - W^T W$ is positive semidefinite if and only if $M$ is, and

$$\nu^T M \nu = \sum_{ab} \nu_a M_{ab} \nu_b \tag{19}$$

$$= \sum_{ab} h_a^t (W^T W)_{ab} h_b^t \nu_a^2 - \nu_a h_a^t (W^T W)_{ab} h_b^t \nu_b \tag{20}$$

$$= \sum_{ab} (W^T W)_{ab} h_a^t h_b^t \left[ \frac{1}{2}\nu_a^2 + \frac{1}{2}\nu_b^2 - \nu_a \nu_b \right] \tag{21}$$

$$= \frac{1}{2} \sum_{ab} (W^T W)_{ab} h_a^t h_b^t (\nu_a - \nu_b)^2 \tag{22}$$

$$\geq 0 \tag{23}$$

■

We can now demonstrate the convergence of Theorem 1:

**Proof of Theorem 1** Replacing $G(h, h^t)$ in Eq. (11) by Eq. (14) results in the update rule:

$$h^{t+1} = h^t - K(h^t)^{-1} \nabla F(h^t) \tag{24}$$

Since Eq. (14) is an auxiliary function, $F$ is nonincreasing under this update rule, according to Lemma 1. Writing the components of this equation explicitly, we obtain

$$h_a^{t+1} = h_a^t \frac{(W^T v)_a}{(W^T W h^t)_a}. \tag{25}$$

By reversing the roles of $W$ and $H$ in Lemma 1 and 2, $F$ can similarly be shown to be nonincreasing under the update rules for $W$. ■

We now consider the following auxiliary function for the divergence cost function:

**Lemma 3** *Define*

$$G(h, h^t) = \sum_i (v_i \log v_i - v_i) + \sum_{ia} W_{ia} h_a \tag{26}$$

$$- \sum_{ia} v_i \frac{W_{ia} h_a^t}{\sum_b W_{ib} h_b^t} \left( \log W_{ia} h_a - \log \frac{W_{ia} h_a^t}{\sum_b W_{ib} h_b^t} \right) \tag{27}$$

*This is an auxiliary function for*

$$F(h) = \sum_i v_i \log \left( \frac{v_i}{\sum_a W_{ia} h_a} \right) - v_i + \sum_a W_{ia} h_a \tag{28}$$

**Proof:** It is straightforward to verify that $G(h, h) = F(h)$. To show that $G(h, h^t) \geq F(h)$, we use convexity of the log function to derive the inequality

$$- \log \sum_a W_{ia} h_a \leq - \sum_a \alpha_a \log \frac{W_{ia} h_a}{\alpha_a} \tag{29}$$

which holds for all nonnegative $\alpha_a$ that sum to unity. Setting

$$\alpha_a = \frac{W_{ia} h_a^t}{\sum_b W_{ib} h_b^t} \tag{30}$$

we obtain

$$- \log \sum_a W_{ia} h_a \leq - \sum_a \frac{W_{ia} h_a^t}{\sum_b W_{ib} h_b^t} \left( \log W_{ia} h_a - \log \frac{W_{ia} h_a^t}{\sum_b W_{ib} h_b^t} \right) \tag{31}$$

From this inequality it follows that $F(h) \leq G(h, h^t)$. ■

Theorem 2 then follows from the application of Lemma 1:

**Proof of Theorem 2:** The minimum of $G(h, h^t)$ with respect to $h$ is determined by setting the gradient to zero:

$$\frac{dG(h, h^t)}{dh_a} = - \sum_i v_i \frac{W_{ia} h_a^t}{\sum_b W_{ib} h_b^t} \frac{1}{h_a} + \sum_i W_{ia} = 0 \tag{32}$$

Thus, the update rule of Eq. (11) takes the form

$$h_a^{t+1} = \frac{h_a^t}{\sum_b W_{kb}} \sum_i \frac{v_i}{\sum_b W_{ib} h_b^t} W_{ia}. \tag{33}$$

Since $G$ is an auxiliary function, $F$ in Eq. (28) is nonincreasing under this update. Rewritten in matrix form, this is equivalent to the update rule in Eq. (5). By reversing the roles of $H$ and $W$, the update rule for $W$ can similarly be shown to be nonincreasing. ■

# 7 Discussion

We have shown that application of the update rules in Eqs. (4) and (5) are guaranteed to find at least locally optimal solutions of Problems 1 and 2, respectively. The convergence proofs rely upon defining an appropriate auxiliary function. We are currently working to generalize these theorems to more complex constraints. The update rules themselves are extremely easy to implement computationally, and will hopefully be utilized by others for a wide variety of applications.

We acknowledge the support of Bell Laboratories. We would also like to thank Carlos Brody, Ken Clarkson, Corinna Cortes, Roland Freund, Linda Kaufman, Yann Le Cun, Sam Roweis, Larry Saul, and Margaret Wright for helpful discussions.

## Footnotes

[1]One can also show that $K - W^T W$ is positive semidefinite by considering the matrix $K^{\frac{1}{2}}(I - K^{-\frac{1}{2}} W^T W K^{-\frac{1}{2}}) K^{\frac{1}{2}}$. Then $\sqrt{h_a^t (W^T W h^t)_a}$ is a positive eigenvector of $K^{-\frac{1}{2}} W^T W K^{-\frac{1}{2}}$ with unity eigenvalue, and application of the Frobenius-Perron theorem shows that Eq. 17 holds.

# References

[1] Jolliffe, IT (1986). *Principal Component Analysis.* New York: Springer-Verlag.

[2] Turk, M & Pentland, A (1991). Eigenfaces for recognition. *J. Cogn. Neurosci.* **3**, 71–86.

[3] Gersho, A & Gray, RM (1992). *Vector Quantization and Signal Compression.* Kluwer Acad. Press.

[4] Lee, DD & Seung, HS. Unsupervised learning by convex and conic coding (1997). *Proceedings of the Conference on Neural Information Processing Systems* **9**, 515–521.

[5] Lee, DD & Seung, HS (1999). Learning the parts of objects by non-negative matrix factorization. *Nature* **401**, 788–791.

[6] Field, DJ (1994). What is the goal of sensory coding? *Neural Comput.* **6**, 559–601.

[7] Foldiak, P & Young, M (1995). Sparse coding in the primate cortex. *The Handbook of Brain Theory and Neural Networks*, 895–898. (MIT Press, Cambridge, MA).

[8] Press, WH, Teukolsky, SA, Vetterling, WT & Flannery, BP (1993). *Numerical recipes: the art of scientific computing.* (Cambridge University Press, Cambridge, England).

[9] Shepp, LA & Vardi, Y (1982). Maximum likelihood reconstruction for emission tomography. *IEEE Trans.* **MI-2**, 113–122.

[10] Richardson, WH (1972). Bayesian-based iterative method of image restoration. *J. Opt. Soc. Am.* **62**, 55–59.

[11] Lucy, LB (1974). An iterative technique for the rectification of observed distributions. *Astron. J.* **74**, 745–754.

[12] Bouman, CA & Sauer, K (1996). A unified approach to statistical tomography using coordinate descent optimization. *IEEE Trans. Image Proc.* **5**, 480–492.

[13] Paatero, P & Tapper, U (1997). Least squares formulation of robust non-negative factor analysis. *Chemometr. Intell. Lab.* **37**, 23–35.

[14] Kivinen, J & Warmuth, M (1997). Additive versus exponentiated gradient updates for linear prediction. *Journal of Information and Computation* **132**, 1–64.

[15] Dempster, AP, Laird, NM & Rubin, DB (1977). Maximum likelihood from incomplete data via the EM algorithm. *J. Royal Stat. Soc.* **39**, 1–38.

[16] Saul, L & Pereira, F (1997). Aggregate and mixed-order Markov models for statistical language processing. In C. Cardie and R. Weischedel (eds). Proceedings of the Second Conference on Empirical Methods in Natural Language Processing, 81–89. ACL Press.
